# Adaptive On-line Learning in Changing Environments

**Noboru Murata, Klaus-Robert Müller, Andreas Ziehe**
GMD-First, Rudower Chaussee 5, 12489 Berlin, Germany
{mura,klaus,ziehe}@first.gmd.de

**Shun-ichi Amari**
Laboratory for Information Representation, RIKEN
Hirosawa 2–1, Wako–shi, Saitama 351–01, Japan
amari@zoo.riken.go.jp

## Abstract

An adaptive on-line algorithm extending the learning of learning idea is proposed and theoretically motivated. Relying only on gradient flow information it can be applied to learning continuous functions or distributions, even when no explicit loss function is given and the Hessian is not available. Its efficiency is demonstrated for a non-stationary blind separation task of acoustic signals.

## 1  Introduction

Neural networks provide powerful tools to capture the structure in data by learning. Often the batch learning paradigm is assumed, where the learner is given all training examples simultaneously and allowed to use them as often as desired. In large practical applications batch learning is often experienced to be rather infeasible and instead on-line learning is employed.

In the on-line learning scenario only one example is given at a time and then discarded after learning. So it is less memory consuming and at the same time it fits well into more natural learning, where the learner receives new information and should adapt to it, without having a large memory for storing old data. On-line learning has been analyzed extensively within the framework of statistics (Robbins & Monro [1951], Amari [1967] and others) and statistical mechanics (see eg. Saad & Solla [1995]). It was shown that on-line learning is asymptotically as effective as batch

learning (cf. Robbins & Monro [1951]). However this only holds, if the appropriate learning rate $\eta$ is chosen. A too large $\eta$ spoils the convergence of learning. In earlier work on dichotomies Sompolinsky et al. [1995] showed the effect on the rate of convergence of the generalization error of a constant, annealed and adaptive learning rate. In particular, the annealed learning rate provides an optimal convergence rate, however it cannot follow changes in the environment. Since on-line learning aims to follow the change of the rule which generated the data, Sompolinsky et al. [1995], Darken & Moody [1991] and Sutton [1992] proposed adaptive learning rates, which learn how to learn. Recently Cichoki et al. [1996] proposed an adaptive on-line learning algorithm for blind separation based on low pass filtering to stabilize learning.

We will extend the reasoning of Sompolinsky et al. in several points: (1) we give an adaptive learning rule for learning continuous functions (section 3) and (2) we consider the case, where no explicit loss function is given and the Hessian cannot be accessed (section 4). This will help us to apply our idea to the problem of on-line blind separation in a changing environment (section 5).

## 2  On-line Learning

Let us consider an infinite sequence of independent examples $(\boldsymbol{x}_1, \boldsymbol{y}_1), (\boldsymbol{x}_2, \boldsymbol{y}_2), \dots$. The purpose of learning is to obtain a network with parameter $\hat{\boldsymbol{w}}$ which can simulate the rule inherent to this data. To this end, the neural network modifies its parameter $\hat{\boldsymbol{w}}_t$ at time $t$ into $\hat{\boldsymbol{w}}_{t+1}$ by using only the next example $(\boldsymbol{x}_{t+1}, \boldsymbol{y}_{t+1})$ given by the rule. We introduce a loss function $l(\boldsymbol{x}, \boldsymbol{y}; \boldsymbol{w})$ to evaluate the performance of the network with parameter $\boldsymbol{w}$. Let $R(\boldsymbol{w}) = \langle l(\boldsymbol{x}, \boldsymbol{y}; \boldsymbol{w}) \rangle$ be the expected loss or the generalization error of the network having parameter $\boldsymbol{w}$, where $\langle \ \rangle$ denotes the average over the distribution of examples $(\boldsymbol{x}, \boldsymbol{y})$. The parameter $\boldsymbol{w}^*$ of the best machine is given by $\boldsymbol{w}^* = \arg\min R(\boldsymbol{w})$. We use the following stochastic gradient descent algorithm (see Amari [1967] and Rumelhart et al. [1986]):

$$\hat{\boldsymbol{w}}_{t+1} = \hat{\boldsymbol{w}}_t - \eta_t C(\hat{\boldsymbol{w}}_t) \frac{\partial}{\partial \boldsymbol{w}} l(\boldsymbol{x}_{t+1}, \boldsymbol{y}_{t+1}; \hat{\boldsymbol{w}}_t), \tag{1}$$

where $\eta_t$ is the learning rate which may depend on $t$ and $C(\hat{\boldsymbol{w}}_t)$ is a positive-definite matrix which may depend on $\hat{\boldsymbol{w}}_t$. The matrix $C$ plays the role of the Riemannian metric tensor of the underlying parameter space $\{\boldsymbol{w}\}$.

When $\eta_t$ is fixed to be equal to a small constant $\eta$, $E[\hat{\boldsymbol{w}}_t]$ converges to $\boldsymbol{w}^*$ and $\mathrm{Var}[\hat{\boldsymbol{w}}_t]$ converges to a non-zero matrix which is order $O(\eta)$. It means that $\hat{\boldsymbol{w}}_t$ fluctuates around $\boldsymbol{w}^*$ (see Amari [1967], Heskes & Kappen [1991]). If $\eta_t = c/t$ (annealed learning rate) $\hat{\boldsymbol{w}}_t$ converges to $\boldsymbol{w}^*$ locally (Sompolinsky et al. [1995]). However when the rule changes over time, an annealed learning rate cannot follow the changes fast enough since $\eta_t = c/t$ is too small.

## 3  Adaptive Learning Rate

The idea of an adaptively changing $\eta_t$ was called learning of the learning rule (Sompolinsky et al. [1995]). In this section we investigate an extension of this idea to differentiable loss functions. Following their algorithm, we consider

$$\hat{\boldsymbol{w}}_{t+1} = \hat{\boldsymbol{w}}_t - \eta_t K^{-1}(\hat{\boldsymbol{w}}_t) \frac{\partial}{\partial \boldsymbol{w}} l(\boldsymbol{x}_{t+1}, \boldsymbol{y}_{t+1}; \hat{\boldsymbol{w}}_t), \tag{2}$$

$$\eta_{t+1} \quad = \quad \eta_t + \alpha \eta_t \left( \beta \left( l(\boldsymbol{x}_{t+1}, \boldsymbol{y}_{t+1}; \hat{w}_t) - \hat{R} \right) - \eta_t \right), \tag{3}$$

where $\alpha$ and $\beta$ are constants, $K(\hat{w}_t)$ is a Hessian matrix of the expected loss function $\partial^2 R(\hat{w}_t)/\partial w \partial w$ and $\hat{R}$ is an estimator of $R(w^*)$. Intuitively speaking, the coefficient $\eta$ in Eq.(3) is controlled by the remaining error. When the error is large, $\eta$ takes a relatively large value. When the error is small, it means that the estimated parameter is close to the optimal parameter; $\eta$ approaches to 0 automatically. However, for the above algorithm all quantities $(K, l, \hat{R})$ have to be accessible which they are certainly not in general. Furthermore $l(\boldsymbol{x}_{t+1}, \boldsymbol{y}_{t+1}; \hat{w}_t) - \hat{R}$ could take negative values. Nevertheless in order to still get an intuition of the learning behaviour, we use the continuous versions of (2) and (3), averaged with respect to the current input-output pair $(\boldsymbol{x}_t, \boldsymbol{y}_t)$ and we omit correlations and variances between the quantities $(\eta_t, w_t, l)$ for the sake of simplicity

$$\frac{d}{dt} w_t = -\eta_t K(w_t)^{-1} \left\langle \frac{\partial}{\partial w} l(\boldsymbol{x}, \boldsymbol{y}; w_t) \right\rangle \text{ and } \frac{d}{dt} \eta_t = \alpha \eta_t \left( \beta \langle l(\boldsymbol{x}, \boldsymbol{y}; w_t) - \hat{R} \rangle - \eta_t \right).$$

Noting that $\langle \partial l(\boldsymbol{x}, \boldsymbol{y}; w^*)/\partial w \rangle = 0$, we have the asymptotic evaluations

$$\left\langle \frac{\partial}{\partial w} l(\boldsymbol{x}, \boldsymbol{y}; w_t) \right\rangle \quad \simeq \quad K^*(w_t - w^*),$$

$$\langle l(\boldsymbol{x}, \boldsymbol{y}; w_t) - \hat{R} \rangle \quad \simeq \quad R(w^*) - \hat{R} + \frac{1}{2}(w_t - w^*)^T K^*(w_t - w^*),$$

with $K^* = \partial^2 R(w^*)/\partial w \partial w$. Assuming $R(w^*) - \hat{R}$ is small and $K(w_t) \simeq K^*$ yields

$$\frac{d}{dt} w_t = -\eta_t(w_t - w^*), \quad \frac{d}{dt} \eta_t = \alpha \eta_t \left( \frac{\beta}{2}(w_t - w^*)^T K^*(w_t - w^*) - \eta_t \right). \tag{4}$$

Introducing the squared error $e_t = \frac{1}{2}(w_t - w^*)^T K^*(w_t - w^*)$, gives rise to

$$\dot{e}_t = -2\eta_t e_t, \quad \dot{\eta}_t = \alpha \beta \eta_t e_t - \alpha \eta_t^2. \tag{5}$$

The behavior of the above equation system is interesting: The origin $(0, 0)$ is its attractor and the basin of attraction has a fractal boundary. Starting from an adequate initial value, it has the solution of the form

$$e_t = \frac{1}{\beta} \left( \frac{1}{2} - \frac{1}{\alpha} \right) \cdot \frac{1}{t} \quad (\alpha > 2), \quad \text{and} \quad \eta_t = \frac{1}{2} \cdot \frac{1}{t}. \tag{6}$$

It is important to note that this $1/t$-convergence rate of the generalization error $e_t$ is the optimal order of any estimator $\hat{w}_t$ converging to $w^*$. So we find that Eq.(4) gives us an on-line learning algorithm which converges with a fast rate. This holds also if the target rule is slowly fluctuating or suddenly changing. The technique to prove convergence was to use the scalar distance in weight space $e_t$. Note also that Eq.(6) holds only within an appropriate parameter range; for small $\eta$ and $w_t - w^*$ correlations and variances between $(\eta_t, w_t, l)$ can no longer be neglected.

## 4 Modification

From the practical point of view (1) the Hessian $K^*$ of the expected loss or (2) the minimum value of the expected loss $\hat{R}$ are in general not known or (3) in some

applications we cannot access the explicit loss function (e.g. blind separation). Let us therefore consider a generalized learning algorithm:

$$\hat{w}_{t+1} = \hat{w}_t - \eta_t f(x_{t+1}, y_{t+1}; \hat{w}_t), \tag{7}$$

where $f$ is a flow which determines the modification when an example $(x_{t+1}, y_{t+1})$ is given. Here we do not assume the existence of a loss function and we only assume that the averaged flow vanishes at the optimal parameter, i.e. $\langle f(x, y; w^*) \rangle = 0$. With a loss function, the flow corresponds to the gradient of the loss. We consider the averaged continuous equation and expand it around the optimal parameter:

$$\frac{d}{dt} w_t = -\eta_t \langle f(x, y; w_t) \rangle \simeq -\eta_t K^*(w_t - w^*), \tag{8}$$

where $K^* = \langle \partial f(x, y; w^*)/\partial w \rangle$. Suppose that we have an eigenvector of the Hessian $K^*$ vector $v$ satisfying $v^T K^* = \lambda v^T$ and let us define

$$\xi_t = \langle v^T f(x, y; w_t) \rangle \simeq v^T K^*(w_t - w^*), \tag{9}$$

then the dynamics of $\xi$ can be approximately represented as

$$\frac{d}{dt} \xi_t = -\lambda \eta_t \xi_t. \tag{10}$$

By using $\xi$, we define a discrete and continuous modification of the rule for $\eta$:

$$\eta_{t+1} = \eta_t + \alpha \eta_t (\beta|\xi_t| - \eta_t) \quad \text{and} \quad \frac{d}{dt}\eta_t = \alpha \eta_t (\beta|\xi_t| - \eta_t). \tag{11}$$

Intuitively $\xi$ corresponds to a 1-dimensional pseudo distance, where the average flow $f$ is projected down to a single direction $v$. The idea is to choose a clever direction such that it is sufficient to observe all dynamics of the flow only along this projection. In this sense the scalar $\xi$ is the simplest obtainable value to observe learning. Noting that $\xi$ is always positive or negative depending on its initial value and $\eta$ can be positive, these two equations (10) and (11) are equivalent to the equation system (5). Therefore their asymptotic solutions are

$$\xi_t = \frac{1}{\beta}\left(\frac{1}{\lambda} - \frac{1}{\alpha}\right) \cdot \frac{1}{t}, \quad \text{and} \quad \eta_t = \frac{1}{\lambda} \cdot \frac{1}{t}. \tag{12}$$

Again similar to the last section we have shown that the algorithm converges properly, however this time without using loss or Hessian. In this algorithm, an important problem is how to get a good projection $v$. Here we assume the following facts and approximate the previous algorithm: **(1)** the minimum eigenvalue of matrix $K^*$ is sufficiently smaller than the second minimum eigenvalue and **(2)** therefore after a large number of iterations, the parameter vector $\hat{w}_t$ will approach from the direction of the minimum eigenvector of $K^*$. Since under these conditions the evolution of the estimated parameter can be thought of as a one-dimensional process, any vector can be used as $v$ except for the vectors which are orthogonal to the minimum eigenvector. The most efficient vector will be the minimum eigenvector itself which can be approximated (for a large number of iterations) by

$$v = \langle f \rangle / \|\langle f \rangle\|,$$

where $\| \; \|$ denotes the $L^2$ norm. Hence we can adopt $\xi = \|\langle f \rangle \|$. Substituting the instantaneous average of the flow by a leaky average, we arrive at

$$\hat{w}_{t+1} = \hat{w}_t - \eta_t f(x_{t+1}, y_{t+1}; \hat{w}_t), \tag{13}$$

$$r_{t+1} = (1-\delta)r_t + \delta f(x_{t+1}, y_{t+1}; \hat{w}_t), \quad (0 < \delta < 1) \tag{14}$$

$$\eta_{t+1} = \eta_t + \alpha \eta_t (\beta \|r_{t+1}\| - \eta_t), \tag{15}$$

where $\delta$ controls the leakiness of the average and $r$ is used as auxiliary variable to calculate the leaky average of the flow $f$. This set of rules is easy to compute. However $\eta$ will now approach a small value because of fluctuations in the estimation of $r$ which depend on the choice of $\alpha, \beta, \gamma$. In practice, to assure the stability of the algorithm, the learning rate in Eq.(13) should be limited to a maximum value $\eta_{\max}$ and a cut-off $\eta_{\min}$ should be imposed.

## 5 Numerical Experiment: an application to blind separation

In the following we will describe the blind separation experiment that we conducted (see eg. Bell & Sejnowski [1995], Jutten & Herault [1991], Molgedey & Schuster [1994] for more details on blind separation). As an example we use the two sun audio files (sampling rate 8kHz): "rooster" ($s_t^1$) and "space music" ($s_t^2$) (see Fig. 1). Both sources are mixed on the computer via $\vec{I}_t = (\mathbb{1} + A)\vec{s}_t$ where $0s < t < 1.25s$ and $3.75s \le t \le 5s$ and $\vec{I}_t = (\mathbb{1} + B)\vec{s}_t$ for $1.25s \le t < 3.75s$, using $A = (0\ 0.9; 0.7\ 0)$ and $B = (0\ 0.8; 0.6\ 0)$ as mixing matrices. So the rule *switches* twice in the given data. The goal is to obtain the sources $\vec{s}_t$ by estimating $\hat{A}$ and $\hat{B}$, given *only* the measured mixed signals $\vec{I}_t$. A change of the mixing is a scenario often encountered in real blind separation tasks, e.g. a speaker turns his head or moves during his utterances. Our on-line algorithm is especially suited to this non-stationary separation task, since adaptation is not limited by the above-discussed generic drawbacks of a constant learning rate as in Bell & Sejnowski [1995], Jutten & Herault [1991], Molgedey & Schuster [1994]. Let $\vec{u}_t$ be the unmixed signals

$$\vec{u}_t = (\mathbb{1} + T_t)^{-1} \vec{I}_t, \tag{16}$$

where $T$ is the estimated mixing matrix. Along the lines of Molgedey & Schuster [1994] we use as modification rule for $T_t$

$$\Delta T_t^{ij} \propto \eta_t f \left( \langle I_t^j u_t^j \rangle, \langle u_t^i u_t^j \rangle, \langle I_t^j u_{t-1}^j \rangle, \langle u_t^i u_{t-1}^j \rangle \right) \propto \eta_t \langle I_t^j u_t^j \rangle \langle u_t^i u_t^j \rangle + \langle I_t^j u_{t-1}^j \rangle \langle u_t^i u_{t-1}^j \rangle,$$

$(i, j = 1, 2, i \ne j)$, where we substitute instantaneous averages with leaky averages

$$\langle I_t^j u_t^j \rangle_{\text{leaky}} = (1-\epsilon) \langle I_{t-1}^j u_{t-1}^j \rangle_{\text{leaky}} + \epsilon I_t^j u_t^j.$$

Note that the necessary ingredients for the flow $f$ in Eq.(13)-(14) are in this case simply the correlations at equal or different times; $\eta_t$ is computed according to Eq.(15). In Fig.2 we observe the results of the simulation (for parameter details, see figure caption). After a short time (t=0.4s) of large $\eta$ and strong fluctuations in $\eta$ the mixing matrix is estimated correctly. Until t=1.25s the learning rate adapts cooling down approximately similar to $1/t$ (cf. Fig. 2c), which was predicted in Eq.(12) in the previous section, i.e. it finds the optimal rate for annealing. At the

point of the switch where simple annealed learning would have failed to adapt to the sudden change, our adaptive rule increases $\eta$ drastically and is able to follow the switch within another 0.4s rsp. 0.1s. Then again, the learning rate is cooled down automatically as intended. Comparing the mixed, original and unmixed signals in Fig.1 confirms the accurate and fast estimate that we already observed in the mixing matrix elements. The same also holds for an acoustic cross check: for a small part of a second both signals are audible, then as time proceeds only one signal, and again after the switches both signals are audible but only for a very short moment . The fading away of the signal is so fast to the listener that it seems that one signal is simply "switched off" by the separation algorithm.

Altogether we found an excellent adaptation behavior of the proposed on-line algorithm, which was also reproduced in other simulation examples omitted here.

## 6 Conclusion

We gave a theoretically motivated adaptive on-line algorithm extending the work of Sompolinsky et al. [1995]. Our algorithm applies to general feed-forward networks and can be used to accelerate learning by the learning about learning strategy in the difficult setting where (a) continuous functions or distributions are to be learned, (b) the Hessian $K$ is not available and (c) no explicit loss function is given. Note, that if an explicit loss function or $K$ is given, this additional information can be incorporated easily, e.g. we can make use of the real gradient otherwise we only rely on the *flow*. Non-stationary blind separation is a typical implementation of the setting (a)-(c) and we use it as an application of the adaptive on-line algorithm in a changing environment. Note that we can apply the learning rate adaptation to most existing blind separation algorithms and thus make them feasible for a non-stationary environment. However, we would like to emphasize that blind separation is just an example for the general adaptive on-line strategy proposed and applications of our algorithm are by no means limited to this scenario. Future work will also consider applications where the rules change more gradually (e.g. *drift*).

## References

Amari, S. (1967) *IEEE Trans. EC* **16**(3):299-307.

Bell, T., Sejnowski, T. (1995) *Neural Comp.* **7**:1129-1159.

Cichocki A., Amari S., Adachi M., Kasprzak W. (1996) Self-Adaptive Neural Networks for Blind Separation of Sources, ISCAS'96 (IEEE), Vol. 2, 157-160.

Darken, C., Moody, J. (1991) in NIPS 3, Morgan Kaufmann, Palo Alto.

Heskes, T.M., Kappen, B. (1991) *Phys. Rev. A* **440**:2718-2726.

Jutten, C., Herault, J. (1991) *Signal Processing* **24**:1-10.

Molgedey, L., Schuster, H.G. (1994) *Phys. Rev. Lett.* **72**(23):3634-3637.

Robbins, H., Monro, S. (1951) *Ann. Math. Statist.*, **22**:400-407.

Rumelhart, D., McClelland, J.L and the PDP Research Group (eds.) (1986), PDP Vol. 1, pp. 318-362, Cambridge, MA: MIT Press.

Saad D., and Solla S. (1995), *Workshop at NIPS'95*, see World–Wide–Web page: http://neural-server.aston.ac.uk/nips95/workshop.html and references therein.

Sompolinsky, H., Barkai, N., Seung, H.S. (1995) in *Neural Networks: The Statistical Mechanics Perspective*, pp. 105-130. Singapore: World Scientific.

Sutton, R.S. (1992) in Proc. 10th nat. conf. on AI, 171-176, MIT Press.

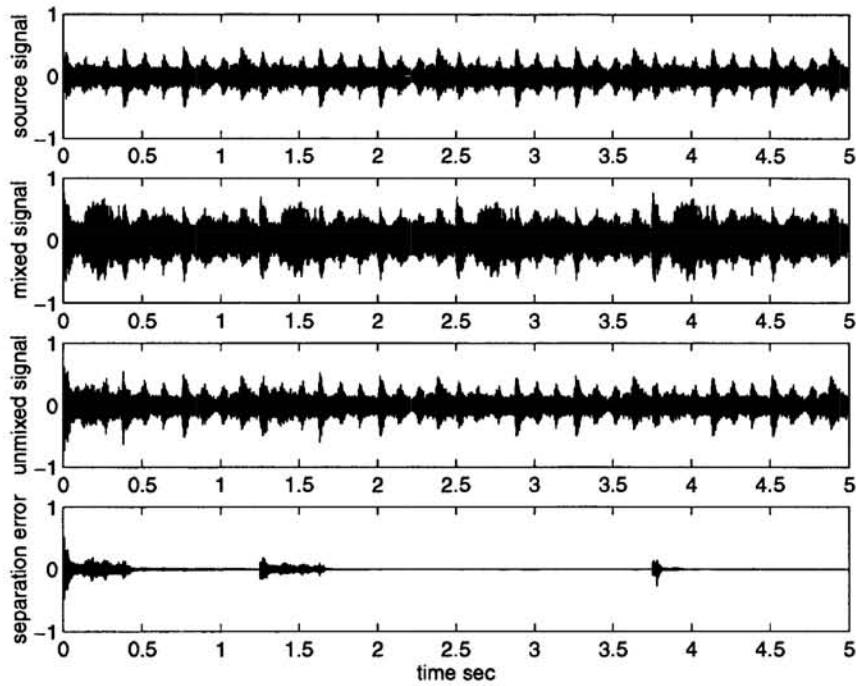

Figure 1: $s_t^2$ "space music", the mixture signal $I_t^2$, the unmixed signal $u_t^2$ and the separation error $u_t^2 - s_t^2$ as functions of time in seconds.

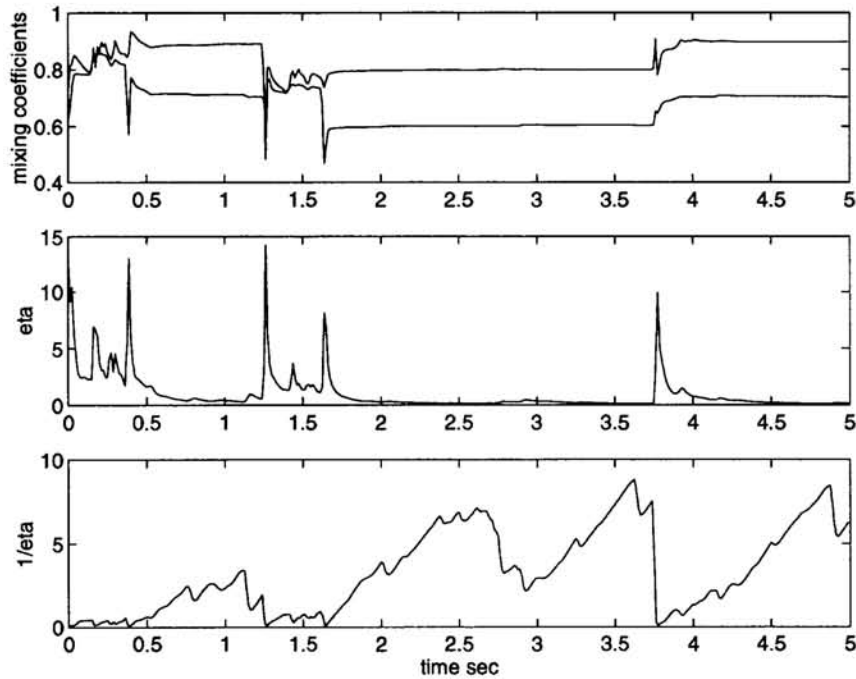

Figure 2: Estimated mixing matrix $T_t$, evolution of the learning rate $\eta_t$ and inverse learning rate $1/\eta_t$ over time. Rule switches (t=1.25s, 3.75s) are clearly observed as drastic changes in $\eta_t$. Asymptotic $1/t$ scaling in $\eta$ amounts to a straight line in $1/\eta_t$. Simulation parameters are $\alpha = 0.002, \beta = 20/\max \|\langle r \rangle\|, \epsilon = \delta = 0.01$. $\max \|\langle r \rangle\|$ denotes the maximal value of the past observations.